# A Surprisingly Simple Approach to
# Generalized Few-Shot Semantic Segmentation

**Tomoya Sakai**
IBM Research - Tokyo
tomoya.sakai2@ibm.com

**Haoxiang Qiu**
IBM Research - Tokyo
haoxiang.qiu@ibm.com

**Takayuki Katsuki**
IBM Research - Tokyo
kats@jp.ibm.com

**Daiki Kimura**
IBM Research - Tokyo
daiki@jp.ibm.com

**Takayuki Osogami**
IBM Research - Tokyo
osogami@jp.ibm.com

**Tadanobu Inoue**
IBM
inouet@jp.ibm.com

## Abstract

The goal of *generalized* few-shot semantic segmentation (GFSS) is to recognize *novel-class* objects through training with a few annotated examples and the *base-class* model that learned the knowledge about the base classes. Unlike the classic few-shot semantic segmentation, GFSS aims to classify pixels into both base and novel classes, meaning it is a more practical setting. Current GFSS methods rely on several techniques such as using combinations of customized modules, carefully designed loss functions, meta-learning, and transductive learning. However, we found that a simple rule and standard supervised learning substantially improve the GFSS performance. In this paper, we propose a simple yet effective method for GFSS that does not use the techniques mentioned above. Also, we theoretically show that our method perfectly maintains the segmentation performance of the base-class model over most of the base classes. Through numerical experiments, we demonstrated the effectiveness of our method. It improved in novel-class segmentation performance in the 1-shot scenario by $6.1\%$ on the PASCAL-$5^i$ dataset, $4.7\%$ on the PASCAL-$10^i$ dataset, and $1.0\%$ on the COCO-$20^i$ dataset. Our code is publicly available at https://github.com/IBM/BCM.

## 1 Introduction

*Semantic segmentation* is a vital task in various visual understanding systems, and the goal is to obtain pixel-level semantic categories [1]. Recent developments in convolutional neural networks [2] and vision transformers [3] have pushed the limits of semantic segmentation. With a large amount of annotated images, we can obtain an accurate model that can recognize objects in the training data. In real-world applications, however, the learned model will encounter *novel-class* objects that are not classified in *base classes*, i.e., classes that are not annotated in the training data.

To solve this problem, *few-shot* semantic segmentation (FSS) aims to recognize novel-class objects with a few annotated images while using the learned model, which has knowledge about the base-class information. Although various FSS methods have been proposed [4–15], FSS only handles novel-class object recognition, which restricts its applicability since base classes will still appear at inference in practice.

*Generalized* FSS (GFSS) aims to recognize both base- and novel-class objects [16] and is regarded as a more practical setting than FSS. Current GFSS methods rely on several techniques such as using combinations of customized modules [16–19], carefully designed loss functions [20], meta-learning [16, 18, 19], and transductive learning [20]. These techniques improved GFSS performance at the

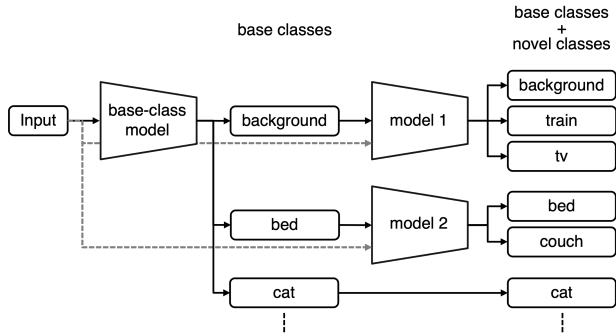

Figure 1: Illustration of *base-class mining* (BCM) with three novel classes: "train", "tv", and "couch". Base-class model first outputs prediction. If prediction is one of chosen base classes, corresponding model outputs prediction. Otherwise, prediction of base-class model is used as it is. Simple rule finds which base class is related to novel classes. Models for novel classes are trained by standard supervised learning.

cost of implementation and computation time. For example, such a customized module is not always supported in the target framework, methods based on meta-learning require several hours to train customized modules, and transductive learning optimizes models during inference, which is not suitable for applications that require quick responses. However, we found that a simple rule and standard supervised learning improve GFSS performance.

In this paper, we propose a simple yet effective GFSS method that does not use the above techniques. As illustrated in Fig. 1, our idea is mining base classes closely related to classifying novel classes. We thus refer to our method as *base-class mining* (BCM). Surprisingly, BCM perfectly maintains the segmentation performance of the base-class model over a subset of base classes. Since maintaining the segmentation performance of the base-class model is critical in GFSS, BCM will be beneficial, especially when a performance difference from the base-class model confuses users.

Our contributions are summarized as follows.

- We propose a simple yet effective GFSS method based on a simple rule and well-known supervised learning techniques, which can be regarded as a strong alternate baseline without transductive learning.
- We theoretically show that the performance of the base-class model for a subset of base classes is perfectly maintained, which is the first theoretical finding about base-class segmentation performances in GFSS, to the best of our knowledge.
- We demonstrated the effectiveness of BCM on the PASCAL-$5^i$, PASCAL-$10^i$, and COCO-$20^i$ datasets. BCM substantially improved novel-class segmentation performance in the 1-shot scenario by $6.1\%$ on PASCAL-$5^i$ and $4.7\%$ on PASCAL-$10^i$.

## 2 Related work

### 2.1 GFSS setting

In GFSS [16], multiple novel classes need to be classified, i.e., multi-class classification, in addition to classifying base classes. This differs from the FSS setting of single novel-class classification, i.e., binary classification.

We consider the *practical* GFSS setting [20], in which the available resources are *few-shot* annotated images for novel classes and the *base-class model* trained using standard learning methods. The existing setting [16] requires annotated base-class samples for training the base-class model and tuning a GFSS model that can recognize both base and novel classes. For example, the number of base classes is 60 (excluding the background) on COCO-$20^i$ case, meaning that we need to collect 300 annotated images for tuning in the 5-shot scenario, other than training the base-class model. Such additional samples for base classes are not necessary in the practical GFSS setting, resulting in using those samples for training the base-class model, which is an advantage of the practical GFSS setting.

### 2.2 GFSS methods

The major challenges with current GFSS methods are i) attaining better recognition performance for novel classes and ii) maintaining the segmentation performance of the base-class model. *Context-aware prototype learning* (CAPL) [16] enhances prototypes with a few annotated images and employs

a balancing mechanism of prototypes for base and novel classes. *Base and meta* (BAM) [14, 18] designed customized modules to learn knowledge from few-shot data and combined predictions of both base and novel classes on the basis of thresholds. POP [17] and PCN [19] use similar approaches. *Distilled information maximization* (DIaM) [20] does not depend on customized modules, instead, uses the *information maximization principle* [21] and designs a loss function on the basis of *knowledge distillation* [22] to preserve base-class knowledge. DIaM uses the transductive learning approach [23], which is not suitable for applications that require quick responses.

Compared with the above methods, BCM does not rely on carefully customized models, various combinations of loss functions, or transductive learning.

### 2.3 Continual semantic segmentation

The GFSS setting relates to another emergence problem setting known as *continual semantic segmentation* (CSS) [24–26] in which the new classes appear in a continual learning manner. The CSS setting is reduced to the GFSS setting if several novel classes appear with a few annotations in a single step. Coincidentally, a previous study [26] empirically found a phenomenon similar to our idea that a novel class is classified as a base class, through qualitative analyses of their method. However, their method was not designed for the few-shot learning setting, meaning that how to improve GFSS performances is unclear. In contrast to their findings, BCM explicitly integrates the idea of the relation between base and novel classes with the architecture. These differences in problem settings and architectures differentiate CSS-based methods from BCM.

## 3 Preliminaries

### 3.1 Problem settings

We consider the following practical GFSS setting [20].

Let $\boldsymbol{X} \in \mathbb{R}^{H \times W \times 3}$ denote an RGB image of height $H$ and width $W$, and $\boldsymbol{Y} \in \mathcal{Y}^{H \times W}$ be its corresponding segmentation map, where $\mathcal{Y} \subset \{0, 1, 2, 3, \dots\}$ is a set of object classes. Let $[\cdot]_j$ indicate the $j$-th element of a matrix, where $j \in \{1, \dots, HW\}$. If $[\boldsymbol{Y}]_j = y$, the object $y$ exists at the $j$-th pixel.

Let $\mathcal{Y}_\mathrm{b}$ and $\mathcal{Y}_\mathrm{n}$ be the sets of *base* and *novel* classes, respectively, such that $\mathcal{Y}_\mathrm{b} \cap \mathcal{Y}_\mathrm{n} = \emptyset$ and $\mathcal{Y}_\mathrm{all} = \mathcal{Y}_\mathrm{b} \cup \mathcal{Y}_\mathrm{n}$. We use the class '0' for background, which is often the case with implementation. For the sake of simplicity, we include the background into $\mathcal{Y}_\mathrm{b}$, e.g., $\mathcal{Y}_\mathrm{b} = \{0, 1, 2, 3\}$ and $\mathcal{Y}_\mathrm{n} = \{4, 5\}$.

We have a learned base-class model $\widehat{g}_\mathrm{b}$, which is trained with a large amount of annotated images by using standard semantic-segmentation methods [1]. Given $\boldsymbol{X}$, $\widehat{g}_\mathrm{b}$ returns a base-class segmentation map $\widehat{\boldsymbol{Y}}_\mathrm{b} \in \mathcal{Y}_\mathrm{b}^{H \times W}$. Similarly to the *practical* GFSS setting [20], we do not assume the customized architecture for $\widehat{g}_\mathrm{b}$, enabling us to easily use cutting-edge foundation models [27].

A $K$-shot dataset contains $K$ examples with its ground-truth mask for each novel class $y \in \mathcal{Y}_\mathrm{n}$, e.g., if $K = 5$ and $|\mathcal{Y}_\mathrm{n}| = 5$, we have 25 annotated images. Note that $K$ examples for base classes are not necessary, as discussed in Sec. 2.1.

Our goal is to obtain the segmentation map $\widehat{\boldsymbol{Y}}_\mathrm{BCM} \in \mathcal{Y}_\mathrm{all}^{H \times W}$ computed using the prediction model for GFSS, denoted as $\widehat{g}_\mathrm{BCM}$.

### 3.2 Evaluation metric

The *mean intersection-over-union* (mIoU) is widely used in reporting the performance of segmentation methods [1]. Let us first define the IoU for a class $y' \in \mathcal{Y}$ as

$$\mathrm{IoU}_{y'}(\boldsymbol{Y}, \widehat{\boldsymbol{Y}}) := \frac{\sum_{j=1}^{HW} \mathrm{I}_{y'}([\boldsymbol{Y}]_j, [\widehat{\boldsymbol{Y}}]_j)}{\sum_{j=1}^{HW} \mathrm{U}_{y'}([\boldsymbol{Y}]_j, [\widehat{\boldsymbol{Y}}]_j)}, \tag{1}$$

where $\mathrm{I}_{y'}(y, \widehat{y}) := \mathbb{I}[y = y'] \cdot \mathbb{I}[\widehat{y} = y']$, $\mathrm{U}_{y'}(y, \widehat{y}) := \mathbb{I}[y = y'] + \mathbb{I}[\widehat{y} = y'] - \mathbb{I}[y = y'] \cdot \mathbb{I}[\widehat{y} = y']$, and $\mathbb{I}[\mathrm{cond}]$ is the indicator function taking 1 if cond is true, 0 otherwise. Here, we consider a single sample for evaluation, but it can be easily extended to multiple samples by adding the summation over

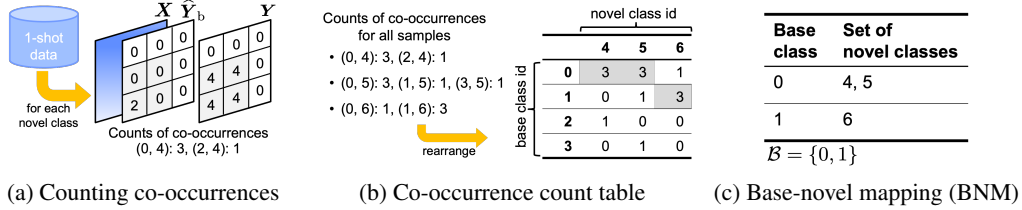

| (a) Counting co-occurrences | (b) Co-occurrence count table | (c) Base-novel mapping (BNM) |

Figure 2: Illustration of BNM creation. For illustration purpose, image of size $3 \times 3$ is used. (a) Count co-occurrences, i.e., (base class, novel class) pairs. There are three $(0, 4)$ and one $(2, 4)$ co-occurrences. (b) Aggregate co-occurrence counts for all samples and create co-occurrence count table. (c) Create BNM from co-occurrence count table, where top-1 strategy finds base class with largest co-occurrences (shaded cell in Fig. 2b) for each novel class.

samples to the numerator and denominator in Eq. (1), respectively. Finally, the mIoU is computed by

$$\text{mIoU}_{\mathcal{Y}}(\boldsymbol{Y}, \widehat{\boldsymbol{Y}}) \coloneqq \frac{1}{|\mathcal{Y}|} \sum_{y' \in \mathcal{Y}} \text{IoU}_{y'}(\boldsymbol{Y}, \widehat{\boldsymbol{Y}}). \tag{2}$$

For example, if $\mathcal{Y}$ is $\mathcal{Y}_{\text{n}}$, it will be the mIoU over novel classes.

# 4 Proposed method

We now present BCM.

## 4.1 Training

Training is divided into two steps: 1) finding the relationship between base and novel classes, and 2) training models for classifying novel classes for each chosen base class.

**Step 1.** We input $\boldsymbol{X}$ into $\widehat{g}_{\text{b}}$ and obtain $\widehat{\boldsymbol{Y}}_{\text{b}}$. For each pixel of the annotated object, we compare $\widehat{\boldsymbol{Y}}_{\text{b}}$ and $\boldsymbol{Y}$ and record co-occurrences of base and novel classes. We then count the co-occurrences, find the top-$s$ co-occurred base class for each novel class, referred to as the *top-$s$ strategy*, and obtain chosen base classes denoted as $\mathcal{B}$. Finally, we construct the mapping from a base class to novel classes, called *base-novel mapping* (BNM). Figure 2 illustrates the creation of BNM with the top-1 strategy from the 1-shot dataset.

**Step 2.** For each chosen base class $\beta \in \mathcal{B}$, we train a model $g_{\beta}$ with the modified $K$-shot dataset where labels are converted into $\beta$ if they are novel classes irrelevant to $\beta$ or the background. Taking the example in Fig. 2c, when $\beta = 1$, the irrelevant novel-class labels '4' and '5' and the background label '0' are replaced with '1'. Then, $\widehat{g}_{\beta=1}$ returns either '1' or '6' as the prediction.

To obtain the learned model $\widehat{g}_{\beta}$, we can use any learning method, such as minimizing the cross-entropy loss or effective losses used in the previous studies.

## 4.2 Inference

Inference is analogous to training. For a test image $\boldsymbol{X}$, we first obtain the base-class prediction $\widehat{\boldsymbol{Y}}_{\text{b}}$. For each pixel $j$, if $[\widehat{\boldsymbol{Y}}_{\text{b}}]_j = \beta$, we then obtain the prediction of the corresponding model $\widehat{g}_{\beta}$ and overwrite $[\widehat{\boldsymbol{Y}}_{\text{b}}]_j$ with the output of $\widehat{g}_{\beta}$. Figure 3 illustrates how we obtain the segmentation map of BCM. We summarize the flow of BCM in Fig. 4.

## 4.3 Preventing catastrophic forgetting

Maintaining the base-class segmentation performance is crucial in GFSS. We theoretically show that BCM perfectly maintains the segmentation performance of most of the base classes without resorting to, e.g., knowledge distillation [22] for training models.

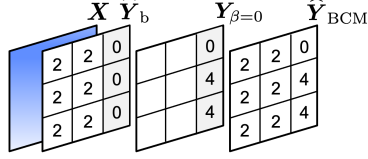

Figure 3: Illustration of inference. $\widehat{Y}_{\rm b}$, $\widehat{Y}_{\beta=0}$, and $\widehat{Y}_{\rm BCM}$ are predictions of $\widehat{g}_{\rm b}$, $\widehat{g}_{\beta=0}$, and BCM, respectively.

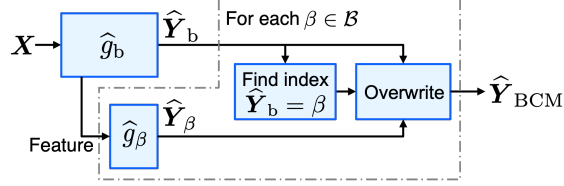

Figure 4: Flow of BCM. For each $\beta \in \mathcal{B}$, BCM finds pixels of $\widehat{Y}_{\rm b} = \beta$ and overwrites $\widehat{Y}_{\rm b}$ with $\widehat{Y}_{\beta}$.

We first formulate the prediction of BCM described in Sec. 4.2. Let $\widehat{y}_{\rm b}$ be the prediction of the base-class model at the $j$-th pixel and $\widehat{y}_{\beta}$ be the prediction of $\widehat{g}_{\beta}$ at the $j$-th pixel. The prediction of BCM at the $j$-th pixel, $\widehat{y}_{\rm BCM}$, is obtained by

$$\widehat{y}_{\rm BCM} = \begin{cases} \widehat{y}_{\rm b} & \text{if } \widehat{y}_{\rm b} \notin \mathcal{B}, \\ \widehat{y}_{\beta=\widehat{y}_{\rm b}} & \text{otherwise.} \end{cases} \tag{3}$$

The above prediction mechanism leads to the following proposition:

**Proposition 4.1.** *Let $\widehat{Y}_{\rm b}$ and $\widehat{Y}_{\rm BCM}$ be the predictions of the base-class model and BCM, respectively. The mIoUs of $\widehat{Y}_{\rm b}$ and $\widehat{Y}_{\rm BCM}$ over $\mathcal{Y}_{\rm b} \setminus \mathcal{B}$ are the same:*

$$\mathrm{mIoU}_{\mathcal{Y}_{\rm b} \setminus \mathcal{B}}(Y, \widehat{Y}_{\rm b}) = \mathrm{mIoU}_{\mathcal{Y}_{\rm b} \setminus \mathcal{B}}(Y, \widehat{Y}_{\rm BCM}). \tag{4}$$

*If $|\mathcal{B}|$ is small, the segmentation performance of most of the base classes is perfectly maintained.*

*Proof.* For any $y' \notin \mathcal{B}$, if $[\widehat{Y}_{\rm b}]_j = y'$, then $[\widehat{Y}_{\rm BCM}]_j = [\widehat{Y}_{\rm b}]_j$ by definition of the BCM prediction. Then, for any $y' \in \mathcal{Y}_{\rm b} \setminus \mathcal{B}$,

$$\mathrm{IoU}_{y'}(Y, \widehat{Y}_{\rm b}) = \mathrm{IoU}_{y'}(Y, \widehat{Y}_{\rm BCM}). \tag{5}$$

Taking the average of $\mathrm{IoU}_{y'}$ over $\mathcal{Y}_{\rm b} \setminus \mathcal{B}$, we obtain Eq. (4). □

Intuitively, since BCM uses the prediction of the base-class model as it is for a subset of base classes, the mIoU over those base classes is the same as $\widehat{g}_{\rm b}$. Proposition 4.1 shows that BCM partially prevents catastrophic forgetting [28, 29]. In our experiments, $|\mathcal{B}|$ tended to be small, resulting in the mIoU over base classes being almost maintained.

### 4.4 Lightweight implementation

Since the size of training data for novel classes is not large in GFSS, training deep neural networks for $\widehat{g}_{\beta}$ is impractical. We thus use the base-class model $\widehat{g}_{\rm b}$ as the feature extractor of $\widehat{g}_{\beta}$ and train linear models as the last layer of $\widehat{g}_{\beta}$ with the $K$-shot data.

To train linear models, we can use off-the-shelf libraries, such as *Scikit-learn* [30], meaning that training time will be fast, compared with the end-to-end training on GPU. Since the number of background pixels is much larger than that of objects of interest, we applied sampling techniques for imbalanced data [31] to training data, such as under-sampling.

Regarding the top-$s$ strategy, we used $s = 1$ from the performance and computation time viewpoint. The effect of $s$ in the top-$s$ strategy is discussed in Sec. 5.6.

### 4.5 Further performance improvement

Since our implementation is to train simple linear models, we can easily use various techniques to improve GFSS performance. We explain two effective and easy-to-use techniques used in our experiments as follows.

**Pre-processing.** We can use *Tukey's ladder of powers transformation* [32], known as the effective transformation in few-shot learning [33]. Specifically, let $\boldsymbol{f}$ be the $d$-dimensional feature vector extracted by $\widehat{g}_{\mathrm{b}}$. Tukey's ladder of powers transformation is defined as

$$\tilde{\boldsymbol{f}} = \begin{cases} \boldsymbol{f}^\tau & \text{if } \tau \neq 0, \\ \log \boldsymbol{f} & \text{otherwise,} \end{cases} \quad (6)$$

where $\tau$ is a hyper-parameter, and the power and logarithm operations are element-wise. When $\tau = 1$, the original feature is used. In a previous study [33], $\tau = 0.5$ was recommended as the default value; thus, we used it in our experiments.

These pre-processed feature vectors are used for $\widehat{g}_\beta$ only since the change in feature representation for $\widehat{g}_{\mathrm{b}}$ without retraining would downgrade performance. Note that to apply similar pre-processing to the existing methods, it is crucial to take into account the adverse effect on total performance.

**Ensemble learning.** We can use ensemble learning [34, 35] to improve GFSS performance. Unlike the existing GFSS methods, the computation time of BCM will be short since training a linear model for $g_\beta$ is lighter than tuning deep neural networks in an end-to-end manner.

We introduce *shot-wise* ensemble learning to few-shot learning when $K > 1$. This involves first preparing multiple $L$-shot datasets ($L \leq K$) by drawing samples from the $K$-shot dataset then aggregating outputs of models trained with the $L$-shot datasets. In our experiments, we split the 5-shot dataset into five 1-shot datasets and obtained six models by using five 1-shot and one 5-shot datasets. In inference, we computed a weighted average of the outputs of the models. The weights can be determined by, e.g., validation data or pre-defined values. In our experiments, we set one for the model with the 1-shot dataset and five for the model with the 5-shot dataset for simplicity.

# 5 Experiments

## 5.1 Setup

**Datasets.** We used three FSS datasets: *PASCAL-$5^i$* [4, 36, 37], *PASCAL-$10^i$* [20, 36, 37], and *COCO-$20^i$* [6, 38]. The PASCAL-$10^i$ dataset was introduced to investigate the impact of increasing the number of novel classes [20].

**Methods for comparison.** We compared BCM with CAPL [16], BAM [14, 18],[1] and DIaM [20]. Note that DIaM was regarded as a *simple* method since it trains the last linear layer only, similarly to the simple methods [39, 40] proposed for few-shot object detection.

**Evaluation.** We report the mIoUs over base and novel classes, referred to as the *Base* and *Novel* scores, respectively, where the background was not included in the Base score, similarly to a previous study [20]. We also report the average of the Base and Novel scores, called the *Mean* score. All reported scores are the average of five independent trials.

**Base-class model.** We used the publicly available pre-trained model for GFSS,[2] *pyramid scene parsing network* (PSPNet) [41] with the pre-trained ResNet-50 backbone [2]. It was trained with labeled data for base classes by using the stochastic gradient descent optimizer with an initial learning rate of $2.5 \times 10^{-4}$, momentum of 0.9, and weight decay of $10^{-4}$. The batch size was 12, and number of epochs was 20 for COCO-$20^i$ and 100 for PASCAL-$5^i$ and PASCAL-$10^i$.

**Detailed implementation.** The implementation of BCM is based on the publicly available DIaM code. We followed the same data-loading and evaluation procedure and replaced the method part with BCM. Specifically, to train novel-class models $g_\beta$ in Sec. 4.1, we used the *logistic regression* in *Scikit-learn* [30],[3] which uses the *L-BFGS-B* [42] method with a line-search strategy as the default solver. The regularization parameter was determined from the five-fold cross-validation from the ten candidates $\{10^{-5}, \ldots, 10^5\}$. The default values were used for the other hyper-parameters.

Table 1: Average mIoU over five trials. Base and Novel represent mIoU scores over base and novel classes, respectively. Mean shows average of Base and Novel scores. Results of comparison methods were obtained from [20]. All methods use ResNet-50 as backbone.

| | | PASCAL-$5^i$ | | | | | |
| | | 1-shot | | | 5-shot | | |
| Method | | Base | Novel | Mean | Base | Novel | Mean |
|---|---|---|---|---|---|---|---|
| CAPL [16] | CVPR'22 | 64.80 | 17.46 | 41.13 | 65.43 | 24.43 | 44.93 |
| BAM [14] | CVPR'22 | **71.60** | 27.49 | 49.55 | **71.60** | 28.96 | 50.28 |
| DIaM [20] | CVPR'23 | 70.89 | 35.11 | 53.00 | 70.85 | 55.31 | 63.08 |
| BCM | (Ours) | 71.15 | **41.24** | **56.20** | 71.23 | **55.36** | **63.29** |
| | | PASCAL-$10^i$ | | | | | |
| CAPL [16] | CVPR'22 | 53.78 | 15.01 | 34.40 | 57.02 | 20.40 | 38.71 |
| BAM [14] | CVPR'22 | 69.02 | 15.48 | 42.25 | 69.18 | 21.51 | 45.35 |
| DIaM [20] | CVPR'23 | **70.26** | 31.29 | 50.77 | **70.25** | 51.89 | 61.07 |
| BCM | (Ours) | 70.07 | **35.94** | **53.01** | 70.12 | **53.49** | **61.81** |
| | | COCO-$20^i$ | | | | | |
| CAPL [16] | CVPR'22 | 43.21 | 7.21 | 25.21 | 43.71 | 11.00 | 27.36 |
| BAM [14] | CVPR'22 | **49.84** | 14.16 | 32.00 | 49.85 | 16.63 | 33.24 |
| DIaM [20] | CVPR'23 | 48.28 | 17.22 | 32.75 | 48.37 | 28.73 | 38.55 |
| BCM | (Ours) | 49.43 | **18.28** | **33.85** | 49.88 | **30.60** | **40.24** |

## 5.2 Main results

Table 1 summarizes the average performance over the five trials for each method in the practical GFSS setting. BCM outperformed the other GFSS methods regarding the Novel and Mean scores. Notably, the Novel scores in the 1-shot PASCAL-$5^i$ and PASCAL-$10^i$ settings substantially improved with BCM. Regarding the Base score, BCM achieved comparable/best performance thanks to it preventing catastrophic forgetting, as discussed in Sec. 4.3. We discuss these results from the viewpoint of our theory in Sec. 5.4.

These results indicate that BCM achieved the best performance without resorting to various techniques used with the other methods, such as meta-learning [43, 44], information maximization principle [21], and transductive learning [23]. The implementation of BCM was to train the final linear layer only, as described in Sec. 4.4, but we can use cutting-edge architectures and training techniques in practice, leading to further performance improvement.

## 5.3 Ablation study

We investigated the effect of pre-processing (Tukey's ladder of powers transformation) and ensemble learning, explained in Sec. 4.5.

Table 2 shows the performance of four variations of BCM, i.e., with and without pre-processing and ensemble learning. Compared with the results in Tab. 1, BCM without data pre-processing and ensemble learning outperformed the other methods in the 1-shot setting, showing that the simple rule and standard supervised learning improved the GFSS performance. The effectiveness of data pre-processing was much higher when the number of novel classes was small (see the PASCAL-$5^i$ and PASCAL-$10^i$ settings). However, the pre-processing decreased this performance slightly in the 1-shot COCO-$20^i$ setting.

Our ensemble-learning approach consistently improved GFSS performance, with a roughly $5\%$ improvement on all datasets. Note that we can use standard ensemble-learning approaches in the 1-shot setting, meaning that further performance improvement is possible in practice.

Table 2: Effect of pre-processing and ensemble learning. 'P' and 'E' denote data pre-processing and ensemble learning, respectively.

| | | PASCAL-$5^i$ | | | | | |
| | | 1-shot | | | 5-shot | | |
| P | E | Base | Novel | Mean | Base | Novel | Mean |
|---|---|---|---|---|---|---|---|
| - | - | **71.16** | 38.13 | 54.65 | **71.23** | 49.83 | 60.53 |
| ✓ | - | 71.15 | **41.24** | **56.20** | **71.23** | 50.83 | 61.03 |
| - | ✓ | - | - | - | **71.23** | 53.53 | 62.38 |
| ✓ | ✓ | - | - | - | **71.23** | **55.36** | **63.29** |
| | | PASCAL-$10^i$ | | | | | |
| - | - | **70.07** | 34.56 | 52.32 | **70.12** | 48.80 | 59.46 |
| ✓ | - | **70.07** | **35.94** | **53.01** | **70.12** | 49.88 | 60.00 |
| - | ✓ | - | - | - | **70.12** | 52.27 | 61.19 |
| ✓ | ✓ | - | - | - | **70.12** | **53.49** | **61.81** |
| | | COCO-$20^i$ | | | | | |
| - | - | **49.48** | 18.03 | 33.76 | 49.88 | 26.73 | 38.30 |
| ✓ | - | 49.43 | **18.28** | **33.85** | 49.86 | 26.76 | 38.31 |
| - | ✓ | - | - | - | **49.90** | 30.48 | 40.19 |
| ✓ | ✓ | - | - | - | 49.88 | **30.60** | **40.24** |

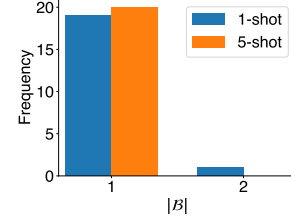

(a) PASCAL-$5^i$

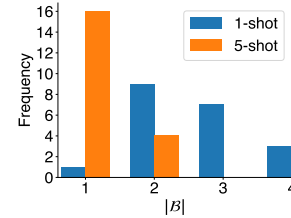

(b) COCO-$20^i$

Figure 5: Frequency of $|\mathcal{B}|$ over four splits and five trials. Size of $\mathcal{B}$ tended to decrease with increasing number of shots.

Table 3: Chosen pairs on COCO-$20^i$ (5-shot)

| Base class | Novel class |
|---|---|
| baseball glove | sports ball |
| bed | couch |
| person | tie<br>baseball glove |

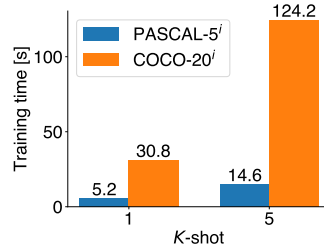

Figure 6: Training time [s] with BCM

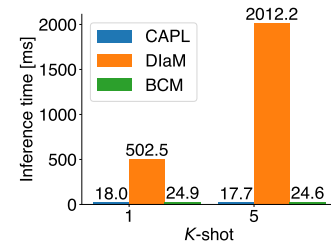

Figure 7: Inference time [ms] on COCO-$20^i$

## 5.4 Number of chosen base classes

Figure 5 shows the frequency of $|\mathcal{B}|$, the number of chosen base classes, over 20 runs (four splits and five trials). Overall, $|\mathcal{B}|$ tended to decrease with the increase in $K$. We hypothesize that noisy pairs appear relatively smaller than frequent pairs when $K = 5$, and the top-1 strategy ignored such a noisy pair. In particular, the median value of the frequency in 5-shot was 1 on PASCAL-$5^i$ and 2 on COCO-$20^i$.

Another observation is that $|\mathcal{B}|$ was much smaller than the number of base classes. For example, the largest $|\mathcal{B}|$ was four in the 5-shot COCO-$20^i$ setting, meaning that less than $7\%(\approx 4/61)$ of base classes (including background) were chosen in the BCM training step. These results indicate that we do not need to prepare $g_\beta$ for many base classes, and training and inference times do not increase rapidly to the number of base classes. Moreover, $|\mathcal{B}|$ tends to be small in practice, so IoUs on most of the base classes are perfectly maintained, as shown in Proposition 4.1.

## 5.5 Which classes were chosen?

We explored which class consists of base- and novel-class pairs in the BNM and recorded pairs in the 5-shot COCO-$20^i$ setting. In most cases, the background was chosen, meaning that the base-class model recognized novel-class objects as the background. Sometimes, the pairs summarized in Tab. 3 were chosen, showing that the related classes were chosen with BCM. The results empirically confirm

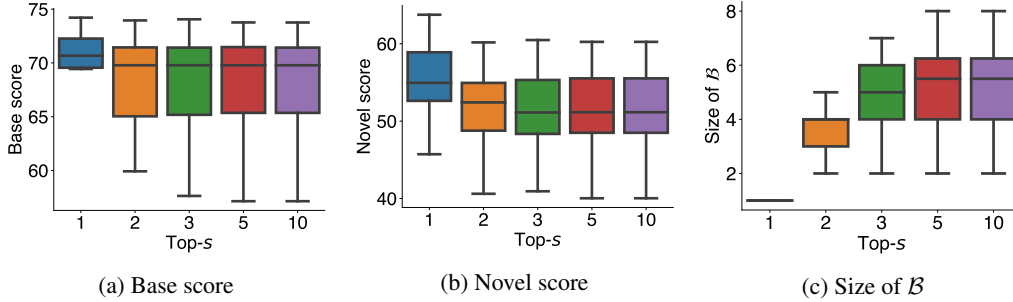

| (a) Base score | (b) Novel score | (c) Size of $\mathcal{B}$ |

Figure 8: Effect of $s$ in top-$s$ strategy in 5-shot PASCAL-$5^i$ setting

our idea that a novel class is classified as the background or a similar base class with the base-class model.

## 5.6   Effect of top-$s$ strategy

We varied $s$ in the top-$s$ strategy and investigated how $s$ affects the segmentation performance and size of $\mathcal{B}$ in the 5-shot PASCAL-$5^i$ setting. The results in other settings can be found in Appendix A.1.

Figures 8a and 8b respectively illustrate the Base and Novel scores with respect to $s \in \{1, 2, 3, 5, 10\}$, showing that the median and minimum scores decreased as $s$ increased.

We also show the size of $\mathcal{B}$ in Fig. 8c. The number of chosen base-classes increased, but it was upper-bounded by a certain value. Even though we increased $s$, the chosen base-classes might be the same. In addition, the increase in $s$ also led to longer training time. In this sense, smaller $s$ is preferable from the perspective of computation time.

In summary, a higher $s$ may result in selecting redundant base classes and cause performance degradation, and larger $|\mathcal{B}|$ requires many models for $g_\beta$, resulting in longer computation time for training. We thus recommend $s = 1$ since it is the best choice based on GFSS performance and computation time.

## 5.7   Computation time

The computation time was measured on a machine equipped with an NVIDIA® V100, 16 CPU cores, and 32GB memory.

**Training time.**   We plot the training time [s] with BCM in Fig. 6. CAPL and DIaM are not shown since CAPL requires hours of training time due to meta-learning, and DIaM, which is based on transductive learning, does not optimize models other than the inference phase. In the 1-shot scenario, training time was less than 1 min. The training time increased as the number of novel classes increased. Although we used ensemble learning in the 5-shot scenario, the training time in the 5-shot COCO-$20^i$ setting was about 2 min. Note that training was done by CPU computation, meaning that further acceleration is expected by GPU computation.

**Inference time.**   Figure 7 shows the inference time [ms] of CAPL, DIaM, and BCM. Since DIaM is based on transductive learning, the inference time was slower than the inductive methods, i.e., CAPL and BCM. BCM requires computations of the novel-class models in addition to that of the base-class model, but the total inference time was comparable to that of the end-to-end model, i.e., CAPL.

Note that BCM has $|\mathcal{B}|$ final linear layers for novel classes, leading to a subtle slowdown when switching the layers, unlike the end-to-end computation of CAPL. Our implementation used CPUs for training the final linear layers, as Scikit-learn is used, unlike CAPL on a GPU. This device difference might be another reason for the slowdown. In practice, a more sophisticated implementation will shorten the gap between CAPL and BCM.

# 6 Conclusion

**Summary.** We presented a simple yet effective GFSS method called BCM. BCM is based on a mapping between base and novel classes and trains novel-class models by simple supervised learning without resorting to meta-learning and the information maximization principle. Since we can use standard supervised learning, training can be done efficiently using off-the-shelf software. We can use the featurizer and base-class model without modifying their weights, enabling us to use cutting-edge public foundation models with BCM. We theoretically showed that the mIoU over most of the base classes is perfectly maintained. Through numerical experiments, we demonstrated the superior performance of BCM method against state-of-the-art GFSS methods.

**Limitations.** BCM has limitations that need to be resolved. First, although the final performance of BCM outperformed the other GFSS methods, the performance improvement in the 5-shot scenario was slight, meaning that there is room for improvement. Second, while BCM perfectly maintains the segmentation performances of most of base-classes, it does not improve such performance using novel-class data. A possible direction to resolve these limitations is to investigate the strategy for creating mapping other than the top-$s$ strategy and use more recent powerful supervised/few-shot learning methods.

# Acknowledgments

The authors would like to thank the anonymous reviewers for their helpful comments and suggestions.

## Footnotes

[1] The details of how we can modify BAM to output multiple novel classes are explained in [20].

[2] DIaM: https://github.com/sinahmr/DIaM

[3] We used an accelerated version of Scikit-learn: https://github.com/intel/scikit-learn-intelex.

# References

[1] Shervin Minaee, Yuri Boykov, Fatih Porikli, Antonio Plaza, Nasser Kehtarnavaz, and Demetri Terzopoulos. Image segmentation using deep learning: A survey. *IEEE TPAMI*, 44(7):3523–3542, 2022. 1, 3

[2] Kaiming He, Xiangyu Zhang, Shaoqing Ren, and Jian Sun. Deep residual learning for image recognition. In *CVPR*, 2016. 1, 6

[3] Alexey Dosovitskiy, Lucas Beyer, Alexander Kolesnikov, Dirk Weissenborn, Xiaohua Zhai, Thomas Unterthiner, Mostafa Dehghani, Matthias Minderer, Georg Heigold, Sylvain Gelly, Jakob Uszkoreit, and Neil Houlsby. An image is worth 16x16 words: Transformers for image recognition at scale. In *ICLR*, 2021. 1

[4] Amirreza Shaban, Shray Bansal, Zhen Liu, Irfan Essa, and Byron Boots. One-shot learning for semantic segmentation. In *BMVC*, 2017. 1, 6

[5] Chi Zhang, Guosheng Lin, Fayao Liu, Rui Yao, and Chunhua Shen. CANet: Class-agnostic segmentation networks with iterative refinement and attentive few-shot learning. In *CVPR*, 2019.

[6] Khoi Nguyen and Sinisa Todorovic. Feature weighting and boosting for few-shot segmentation. In *ICCV*, 2019. 6

[7] Kaixin Wang, Jun Hao Liew, Yingtian Zou, Daquan Zhou, and Jiashi Feng. PANet: Few-shot image semantic segmentation with prototype alignment. In *ICCV*, 2019.

[8] Boyu Yang, Chang Liu, Bohao Li, Jianbin Jiao, and Qixiang Ye. Prototype mixture models for few-shot semantic segmentation. In *ECCV*, pages 763–778, 2020.

[9] Haochen Wang, Xudong Zhang, Yutao Hu, Yandan Yang, Xianbin Cao, and Xiantong Zhen. Few-shot semantic segmentation with democratic attention networks. In *ECCV*, pages 730–746, 2020.

[10] Malik Boudiaf, Hoel Kervadec, Ziko Imtiaz Masud, Pablo Piantanida, Ismail Ben Ayed, and Jose Dolz. Few-shot segmentation without meta-learning: A good transductive inference is all you need? In *CVPR*, pages 13979–13988, 2021.

[11] Bingfeng Zhang, Jimin Xiao, and Terry Qin. Self-guided and cross-guided learning for few-shot segmentation. In *CVPR*, pages 8312–8321, 2021.

[12] Zhihe Lu, Sen He, Xiatian Zhu, Li Zhang, Yi-Zhe Song, and Tao Xiang. Simpler is better: Few-shot semantic segmentation with classifier weight transformer. In *ICCV*, pages 8741–8750, 2021.

[13] Zhuotao Tian, Hengshuang Zhao, Michelle Shu, Zhicheng Yang, Ruiyu Li, and Jiaya Jia. Prior guided feature enrichment network for few-shot segmentation. *IEEE TPAMI*, 44(2):1050–1065, 2022.

[14] Chunbo Lang, Gong Cheng, Binfei Tu, and Junwei Han. Learning what not to segment: A new perspective on few-shot segmentation. In *CVPR*, pages 8057–8067, 2022. 3, 6, 7

[15] Nico Catalano and Matteo Matteucci. Few shot semantic segmentation: a review of methodologies and open challenges, 2023. 1

[16] Zhuotao Tian, Xin Lai, Li Jiang, Shu Liu, Michelle Shu, Hengshuang Zhao, and Jiaya Jia. Generalized few-shot semantic segmentation. In *CVPR*, pages 11563–11572, 2022. 1, 2, 6, 7

[17] Sun-Ao Liu, Yiheng Zhang, Zhaofan Qiu, Hongtao Xie, Yongdong Zhang, and Ting Yao. Learning orthogonal prototypes for generalized few-shot semantic segmentation. In *CVPR*, pages 11319–11328, 2023. 3

[18] Chunbo Lang, Gong Cheng, Binfei Tu, Chao Li, and Junwei Han. Base and meta: A new perspective on few-shot segmentation. *IEEE TPAMI*, 45(9):10669–10686, 2023. 1, 3, 6

[19] Zhihe Lu, Sen He, Da Li, Yi-Zhe Song, and Tao Xiang. Prediction calibration for generalized few-shot semantic segmentation. *IEEE TPAMI*, 32:3311–3323, 2023. 1, 3

[20] Sina Hajimiri, Malik Boudiaf, Ismail Ben Ayed, and Jose Dolz. A strong baseline for generalized few-shot semantic segmentation. In *CVPR*, pages 11269–11278, 2023. 1, 2, 3, 6, 7

[21] R. Linsker. Self-organization in a perceptual network. *Computer*, 21(3):105–117, 1988. 3, 7

[22] Geoffrey Hinton, Oriol Vinyals, and Jeff Dean. Distilling the knowledge in a neural network. In *NIPS Deep Learning and Representation Learning Workshop*, 2015. 3, 4

[23] Olivier Chapelle, Bernhard Schlkopf, and Alexander Zien. *Semi-Supervised Learning*. The MIT Press, 2010. 3, 7

[24] Fabio Cermelli, Massimiliano Mancini, Samuel Rota Bulo, Elisa Ricci, and Barbara Caputo. Modeling the background for incremental learning in semantic segmentation. In *CVPR*, 2020. 3

[25] Ze Yang, Ruibo Li, Evan Ling, Chi Zhang, Yiming Wang, Dezhao Huang, Keng Teck Ma, Minhoe Hur, and Guosheng Lin. Label-guided knowledge distillation for continual semantic segmentation on 2D images and 3D point clouds. In *ICCV*, pages 18601–18612, 2023.

[26] Beomyoung Kim, Joonsang Yu, and Sung Ju Hwang. ECLIPSE: Efficient continual learning in panoptic segmentation with visual prompt tuning. In *CVPR*, pages 3346–3356, 2024. 3

[27] Rishi Bommasani, Drew A. Hudson, Ehsan Adeli, Russ Altman, Simran Arora, Sydney von Arx, Michael S. Bernstein, Jeannette Bohg, Antoine Bosselut, Emma Brunskill, Erik Brynjolfsson, Shyamal Buch, Dallas Card, Rodrigo Castellon, Niladri Chatterji, Annie Chen, Kathleen Creel, Jared Quincy Davis, Dora Demszky, Chris Donahue, Moussa Doumbouya, Esin Durmus, Stefano Ermon, John Etchemendy, Kawin Ethayarajh, Li Fei-Fei, Chelsea Finn, Trevor Gale, Lauren Gillespie, Karan Goel, Noah Goodman, Shelby Grossman, Neel Guha, Tatsunori Hashimoto, Peter Henderson, John Hewitt, Daniel E. Ho, Jenny Hong, Kyle Hsu, Jing Huang, Thomas Icard, Saahil Jain, Dan Jurafsky, Pratyusha Kalluri, Siddharth Karamcheti, Geoff Keeling, Fereshte Khani, Omar Khattab, Pang Wei Koh, Mark Krass, Ranjay Krishna, Rohith Kuditipudi, Ananya Kumar, Faisal Ladhak, Mina Lee, Tony Lee, Jure Leskovec, Isabelle Levent, Xiang Lisa Li, Xuechen Li, Tengyu Ma, Ali Malik, Christopher D. Manning, Suvir Mirchandani, Eric Mitchell, Zanele Munyikwa, Suraj Nair, Avanika Narayan, Deepak Narayanan, Ben Newman, Allen Nie, Juan Carlos Niebles, Hamed Nilforoshan, Julian Nyarko, Giray Ogut, Laurel Orr, Isabel Papadimitriou, Joon Sung Park, Chris Piech, Eva Portelance, Christopher Potts, Aditi Raghunathan, Rob Reich, Hongyu Ren, Frieda Rong, Yusuf Roohani, Camilo Ruiz, Jack Ryan, Christopher Ré, Dorsa Sadigh, Shiori Sagawa, Keshav Santhanam, Andy Shih, Krishnan Srinivasan, Alex Tamkin, Rohan Taori, Armin W. Thomas, Florian Tramèr, Rose E. Wang, William Wang, Bohan Wu, Jiajun Wu, Yuhuai Wu, Sang Michael Xie, Michihiro Yasunaga, Jiaxuan You, Matei Zaharia, Michael Zhang, Tianyi Zhang, Xikun Zhang, Yuhui Zhang, Lucia Zheng, Kaitlyn Zhou, and Percy Liang. On the opportunities and risks of foundation models, 2022. 3

[28] Michael McCloskey and Neal J. Cohen. Catastrophic interference in connectionist networks: The sequential learning problem. In *Psychology of Learning and Motivation*, volume 24, pages 109–165. Academic Press, 1989. 5

[29] James Kirkpatrick, Razvan Pascanu, Neil Rabinowitz, Joel Veness, Guillaume Desjardins, Andrei A. Rusu, Kieran Milan, John Quan, Tiago Ramalho, Agnieszka Grabska-Barwinska, Demis Hassabis, Claudia Clopath, Dharshan Kumaran, and Raia Hadsell. Overcoming catastrophic forgetting in neural networks. *Proceedings of the National Academy of Sciences*, 114(13):3521–3526, 2017. 5

[30] Fabian Pedregosa, Gaël Varoquaux, Alexandre Gramfort, Vincent Michel, Bertrand Thirion, Olivier Grisel, Mathieu Blondel, Peter Prettenhofer, Ron Weiss, Vincent Dubourg, Jake Vanderplas, Alexandre Passos, David Cournapeau, Matthieu Brucher, Matthieu Perrot, and Édouard Duchesnay. Scikit-learn: Machine learning in Python. *JMLR*, 12:2825–2830, 2011. 5, 6

[31] Nitesh V. Chawla. *Data Mining for Imbalanced Datasets: An Overview*, pages 853–867. Springer, 2005. 5

[32] John Wilder Tukey. *Exploratory Data Analysis*. Addison-Wesley Publishing Company, 1977. 6

[33] Shuo Yang, Lu Liu, and Min Xu. Free lunch for few-shot learning: Distribution calibration. In *ICLR*, 2021. 6

[34] Trevor Hastie, Robert Tibshirani, and Jerome Friedman. *The Elements of Statistical Learning*. Springer New York Inc., 2009. 6

[35] Zhi-Hua Zhou. *Ensemble Methods: Foundations and Algorithms*. Chapman & Hall/CRC, 2012. 6

[36] Mark Everingham, SM Eslami, Luc Van Gool, Christopher KI Williams, John Winn, and Andrew Zisserman. The PASCAL visual object classes challenge: A retrospective. *IJCV*, 111(1):98–136, 2015. 6

[37] Bharath Hariharan, Pablo Arbeláez, Ross Girshick, and Jitendra Malik. Simultaneous detection and segmentation. In *ECCV*, 2014. 6

[38] Tsung-Yi Lin, Michael Maire, Serge Belongie, James Hays, Pietro Perona, Deva Ramanan, Piotr Dollár, and C Lawrence Zitnick. Microsoft COCO: Common objects in context. In *ECCV*, pages 740–755, 2014. 6

[39] Xin Wang, Thomas Huang, Joseph Gonzalez, Trevor Darrell, and Fisher Yu. Frustratingly simple few-shot object detection. In *ICML*, volume 119, pages 9919–9928, 2020. 6

[40] Ze Yang, Chi Zhang, Ruibo Li, Yi Xu, and Guosheng Lin. Efficient few-shot object detection via knowledge inheritance. *IEEE Transactions on Image Processing*, 32:321–334, 2023. 6

[41] Hengshuang Zhao, Jianping Shi, Xiaojuan Qi, Xiaogang Wang, and Jiaya Jia. Pyramid scene parsing network. In *CVPR*, 2017. 6

[42] Ciyou Zhu, Richard H. Byrd, Peihuang Lu, and Jorge Nocedal. Algorithm 778: L-BFGS-B: Fortran subroutines for large-scale bound-constrained optimization. *ACM Trans. Math. Softw.*, 23(4):550–560, 1997. 6

[43] Jurgen Schmidhuber. Evolutionary principles in self-referential learning. On learning now to learn: The meta-meta-meta...-hook. Diploma thesis, Technische Universitat Munchen, Germany, 1987. 7

[44] Chelsea Finn, Pieter Abbeel, and Sergey Levine. Model-agnostic meta-learning for fast adaptation of deep networks. In *ICML*, volume 70, pages 1126–1135, 2017. 7

# A    Appendix

## A.1    Effect of top-$s$ strategy

Figs. 9 and 10 show the effect of $s$ in the top-$s$ strategy in the 1-shot PASCAL-$5^i$ and COCO-$20^i$ settings, respectively. We can observe a similar tendency to that discussed in Sec. 5.6.

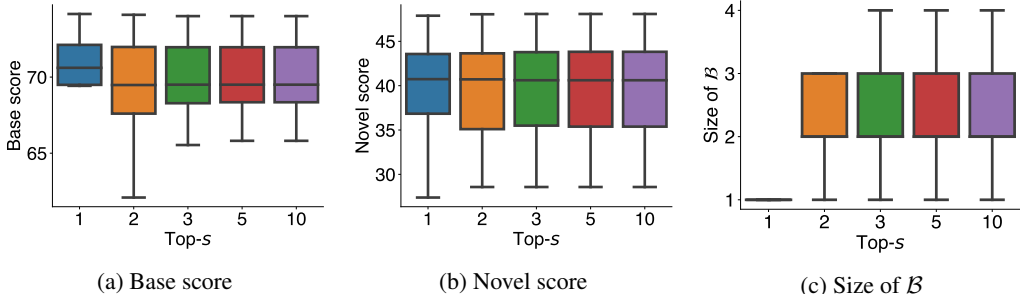

(a) Base score            (b) Novel score            (c) Size of $\mathcal{B}$

Figure 9: Effect of $s$ in top-$s$ strategy in 1-shot PASCAL-$5^i$ setting

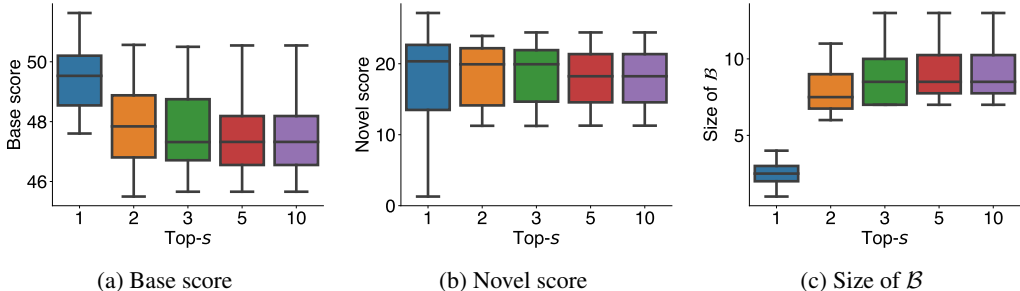

(a) Base score            (b) Novel score            (c) Size of $\mathcal{B}$

Figure 10: Effect of $s$ in top-$s$ strategy in 1-shot COCO-$20^i$ setting

## A.2    Example when $s > 1$

We explain obtaining the prediction when $s > 1$.

Even when $s > 1$, the inference procedure works as explained in Sec. 4.2 since a *single* base class is mapped to multiple novel classes in BNM. We show the case when $s = 2$ with a tiny example: the base classes are '0' and '1', and the novel class is '2'. Suppose that we have the following BNM when $s = 2$ in Tab. 4. This table shows when the base class '0' is mapped to the novel class '2' and '1' is also mapped to '2'. In this case, we have the two models: $g_{\beta=0}$ returns '0' or '2', and $g_{\beta=1}$ returns '1' or '2'.

Table 4: Example BNM when $s = 2$

| Base class | Set of novel classes |
|:----------:|:--------------------:|
| 0 | 2 |
| 1 | 2 |

For each pixel, the base-class model outputs either '0' or '1'. We then compute the prediction of the corresponding model and overwrite it. Since BCM does not need to overwrite the same pixel multiple times, we can straightforwardly combine predictions of $g_\beta$ for the final prediction.

## A.3    Broader impact

The idea behind BCM will positively affect future studies on few-shot learning. Our future work will lead to more powerful visual understanding systems. Regarding negative societal impact, we expect BCM will not have a direct path to harmful applications. However, harmful actors may maliciously use visual understanding systems. To prevent such a malicious use of technology, we need to pay attention to events in our society.

